# Stochastic Dynamics of Three–State Neural Networks

**Toru Ohira**
Sony Computer Science Laboratory
3-14-13 Higashi-gotanda,
Tokyo 141, Japan
ohira@csl.sony.co.jp

**Jack D. Cowan**
Depts. of Mathematics and Neurology
University of Chicago
Chicago, IL 60637
cowan@synapse.uchicago.edu

## Abstract

We present here an analysis of the stochastic neurodynamics of a neural network composed of three–state neurons described by a master equation. An outer–product representation of the master equation is employed. In this representation, an extension of the analysis from two to three–state neurons is easily performed. We apply this formalism with approximation schemes to a simple three–state network and compare the results with Monte Carlo simulations.

## 1  INTRODUCTION

Studies of single neurons or networks under the influence of noise have been a continuing item in neural network modelling. In particular, the analogy with spin systems at finite temprature has produced many important results on networks of two–state neurons. However, studies of networks of three–state neurons have been rather limited (Meunier, Hansel and Verga, 1989). A master equation was introduced by Cowan (1991) to study stochastic neural networks. The equation uses the formalism of "second quantization" for classical many–body systems (Doi, 1976a; Grassberger and Scheunert, 1980), and was used to study networks of of two–state neurons (Ohira and Cowan, 1993, 1994). In this paper, we reformulate the master equation using an outer–product representation of operators and extend our previous analysis to networks of three–state neurons. A hierarchy of moment equations for such networks is derived and approximation schemes are used to obtain equa-

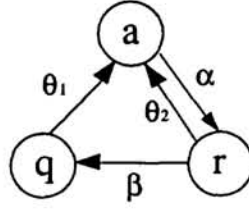

Figure 1: Transition rates for a three–state neuron.

tions for the macroscopic activities of model networks. We compare the behavior of the solutions of these equations with Monte Carlo simulations.

## 2   THE BASIC NEURAL MODEL

We first introduce the network described by the master equation. In this network (Cowan, 1991), neurons at each site, say the $i$th site, are assumed to cycle through three states: "quiescent", "activated" and "refractory", labelled '$q_i$', '$a_i$', and '$r_i$' respectively. We consider four transitions: $q \to a$, $r \to a$, $a \to r$, and $r \to q$. Two of these, $q \to a$ and $r \to a$, are functions of the neural input current. We assume these are smoothly increasing functions of the input current and denoted them by $\theta_1$, and $\theta_2$. The other two transition rates, $a \to r$, and $r \to q$, are defined as constants $\alpha$ and $\beta$. The resulting stochastic transition scheme is shown in Figure 1. We assume that these transition rates depend only on the current state of the network and not on past states, and that all neural state transitions are asynchronous. This Markovian assumption is essential to the master equation description of this model.

We represent the state of each neuron by three–dimensional basis vectors using the Dirac notation $|a_i>$, $|r_i>$ and $|q_i>$. They correspond, in more standard vector notation, to:

$$|q_i> = \begin{pmatrix} 0 \\ 0 \\ 1 \end{pmatrix}_i, \quad |a_i> = \begin{pmatrix} 1 \\ 0 \\ 0 \end{pmatrix}_i, \quad |r_i> = \begin{pmatrix} 0 \\ 1 \\ 0 \end{pmatrix}_i \tag{1}$$

We define the inner product of these states as

$$< a_i|a_i> =< q_i|q_i> =< r_i|r_i> = 1, \tag{2}$$

$$< q_i|a_i> =< a_i|q_i> =< r_i|a_i> =< a_i|r_i> =< r_i|q_i> =< q_i|r_i> = 0. \tag{3}$$

Let the states (or configurations) of a network be represented by $\{|\Omega>\}$, the direct product space of each neuron in the network.

$$|\Omega> = |v_1> |v_2> \ldots |v_N>, \qquad v_i = a_i, \ r_i, or \ q_i. \tag{4}$$

Let P[$\Omega$, t] be the probability of finding the network in a particular state $\Omega$ at time t. We introduce the "neural state vector" for $N$ neurons in a network as

$$|\Phi(t)> = \sum_{\{\Omega\}} P[\Omega, \ t]|\ \Omega>, \tag{5}$$

where the sum is taken over all possible network states.

With these definitions, we can write the master equation for a network with the transition rates shown in Figure 1, using the outer–product representations of operators (Sakurai, 1985). For example:

$$|a_i><q_i| = \begin{pmatrix} 0 & 0 & 1 \\ 0 & 0 & 0 \\ 0 & 0 & 0 \end{pmatrix}_i \tag{6}$$

The master equation then takes the form of an evolution equation:

$$-\frac{\partial}{\partial t}|\Phi(t)> = L|\Phi(t)> \tag{7}$$

with the network "Liouvillian" $L$ given by:

$$L = \alpha \sum_{i=1}^{N}(|a_i\rangle\langle a_i| - |r_i\rangle\langle a_i|) + \sum_{i=1}^{N}(|r_i\rangle\langle r_i| - |a_i\rangle\langle r_i|)\theta_2(\frac{1}{\bar{n}}\sum_{j=1}^{N}w_{ij}|a_j\rangle\langle a_j|)$$

$$+\beta \sum_{i=1}^{N}(|r_i\rangle\langle r_i| - |q_i\rangle\langle r_i|) + \sum_{i=1}^{N}(|q_i\rangle\langle q_i| - |a_i\rangle\langle q_i|)\theta_1(\frac{1}{\bar{n}}\sum_{j=1}^{N}w_{ij}|a_j\rangle\langle a_j|). \tag{8}$$

where $\bar{n}$ is an average number of connections to each neuron, and $w_{ij}$ is the "weight" from the $j$th to the $i$th neuron. Thus the weights are normalized with respect to the average number $\bar{n}$ of connections per neuron.

The master equation given here is the same as the one introduced by Cowan using Gell-Mann matrices (Cowan, 1991). However, we note that with the outer–product representation, we can extend the description from two to three–state neurons simply by including one more basis vector.

In analogy with the analysis of two–state neurons, we introduce the state vector:

$$\langle \vec{a} \; \vec{r} \; \vec{q}| = \prod_{i=1}^{N}(q_i\langle q_i| + r_i\langle r_i| + a_i\langle a_i|). \tag{9}$$

where the product is taken as a direct product, and $a_i$, $r_i$, and $q_i$ are parameters. We also introduce the point moments $\ll a_i(t)\gg$, $\ll q_i(t)\gg$, and $\ll r_i(t)\gg$ as the probability that the $i$th neuron is active, quiescent, and refractory respectively, at time $t$. Similarly, we can define the multiple moment, for example, $\ll a_i q_j r_k \ldots (t)\gg$ as the probability that the $i$th neuron is active, the $j$th neuron is quiescent, the $k$th neuron is refractory and so on at time $t$. Then, it can be shown that they are given by:

$$\ll s_i s_j s_k \ldots (t)\gg = \langle \vec{a} = \vec{r} = \vec{q} = 1|s_i\rangle\langle s_i| \otimes |s_j\rangle\langle s_j| \otimes |s_k\rangle\langle s_k| \ldots |\Phi(t)\rangle,$$

$$s = a, \; r, q \tag{10}$$

For example,

$$\ll r_i q_j a_k(t)\gg = \langle \vec{a} = \vec{r} = \vec{q} = 1|r_i\rangle\langle r_i| \otimes |q_j\rangle\langle q_j| \otimes |a_k\rangle\langle a_k|\Phi(t)\rangle \tag{11}$$

We note the following relations,

$$\ll a_i(t)\gg + \ll q_i(t)\gg + \ll r_i(t)\gg = 1 \tag{12}$$

and

$$\ll a_i^2(t)\gg = \ll a_i(t)\gg, \quad \ll r_i^2(t)\gg = \ll r_i(t)\gg, \quad \ll q_i^2(t)\gg = \ll q_i(t)\gg. \tag{13}$$

# 3   THE HIERARCHY OF MOMENT EQUATIONS

We can now obtain an equation of motion for the moments. As is typical in the case of many–body problems, we obtain an analogue of the BBGKY hierarchy of equations (Doi, 1976b). This can be done by using the definition of moments, the master equation, and the a-r-q state vector. We show the hierarchy up to the second order:

$$-\frac{\partial}{\partial t}\ll a_i\gg = \alpha\ll a_i\gg - \ll r_i\theta_2(\frac{1}{n}\sum_{j=1}^{N}w_{ij}a_j)\gg - \ll q_i\theta_1(\frac{1}{n}\sum_{j=1}^{N}w_{ij}a_j)\gg \quad (14)$$

$$-\frac{\partial}{\partial t}\ll r_i\gg = -\alpha\ll a_i\gg + \beta\ll r_i\gg + \ll r_i\theta_2(\frac{1}{n}\sum_{j=1}^{N}w_{ij}a_j)\gg \quad (15)$$

$$-\frac{\partial}{\partial t}\ll q_i\gg = -\beta\ll r_i\gg + \ll q_i\theta_1(\frac{1}{n}\sum_{j=1}^{N}w_{ij}a_j)\gg \quad (16)$$

$$-\frac{\partial}{\partial t}\ll a_ia_j\gg = 2\alpha\ll a_ia_j\gg - \ll r_ia_j\theta_2(\frac{1}{n}\sum_{k=1}^{N}w_{ik}a_k)\gg - \ll a_ir_j\theta_2(\frac{1}{n}\sum_{k=1}^{N}w_{jk}a_k)\gg$$
$$-\ll q_ia_j\theta_1(\frac{1}{n}\sum_{k=1}^{N}w_{ik}a_k)\gg - \ll a_iq_j\theta_1(\frac{1}{n}\sum_{k=1}^{N}w_{jk}a_k)\gg \quad (17)$$

$$-\frac{\partial}{\partial t}\ll r_ir_j\gg = -\alpha(\ll r_ia_j\gg + \ll a_ir_j\gg) + 2\beta\ll r_ir_j\gg$$
$$+\ll r_ir_j\theta_2(\frac{1}{n}\sum_{k=1}^{N}w_{ik}a_k)\gg + \ll r_ir_j\theta_2(\frac{1}{n}\sum_{k=1}^{N}w_{jk}a_k)\gg \quad (18)$$

$$-\frac{\partial}{\partial t}\ll a_ir_j\gg = -\alpha(\ll a_ia_j\gg - \ll a_ir_j\gg) + \beta\ll a_ir_j\gg + \ll a_ir_j\theta_2(\frac{1}{n}\sum_{k=1}^{N}w_{ik}a_k)\gg$$
$$-\ll r_ir_j\theta_2(\frac{1}{n}\sum_{k=1}^{N}w_{ik}a_k)\gg - \ll q_ir_j\theta_1(\frac{1}{n}\sum_{k=1}^{N}w_{ik}a_k)\gg \quad (19)$$

We note that since

$$\ll a_i\gg + \ll r_i\gg + \ll q_i\gg = 1, \quad (20)$$

one of the parameters can be eliminated. We also note that the equations are coupled into higher orders in this hierarchy. This leads to a need for approximation schemes which can terminate the hierarchy at an appropriate order.

In the following, we introduce first and the second moment level approximation schemes. For simplicity, we consider the special case in which $\theta_1$ and $\theta_2$ are linear and equal.

$$\theta_1(\frac{1}{n}\sum_{j=1}^{N}w_{ij}\ll a_j\gg) = \theta_2(\frac{1}{n}\sum_{j=1}^{N}w_{ij}\ll a_j\gg) = \frac{1}{n}\sum_{j=1}^{N}w_{ij}\ll a_j\gg \quad (21)$$

With the above simplication the first moment (mean field) approximation leads to:

$$-\frac{\partial}{\partial t}\ll a_i \gg = \alpha \ll a_i \gg - \overline{w_i}(\ll r_i \gg + \ll q_i \gg) \tag{22}$$

$$-\frac{\partial}{\partial t}\ll r_i \gg = -\alpha \ll a_i \gg + \beta \ll r_i \gg + \overline{w_i}\ll r_i \gg, \tag{23}$$

$$-\frac{\partial}{\partial t}\ll q_i \gg = -\beta \ll r_i \gg + \overline{w_i}\ll q_i \gg, \tag{24}$$

where

$$\overline{w_l} = \frac{1}{n}\sum_{k=1}^{N} w_{lk}\ll a_k \gg. \tag{25}$$

We also obtain the second moment approximation as:

$$-\frac{\partial}{\partial t}\ll a_i \gg = \alpha \ll a_i \gg - \frac{1}{n}\sum_{j=1}^{N} w_{ij}(\ll q_i a_j \gg + \ll r_i a_j \gg), \tag{26}$$

$$-\frac{\partial}{\partial t}\ll r_i \gg = -\alpha \ll a_i \gg + \beta \ll r_i \gg + \frac{1}{n}\sum_{j=1}^{N} w_{ij}\ll r_i a_j \gg, \tag{27}$$

$$-\frac{\partial}{\partial t}\ll q_i \gg = -\beta \ll r_i \gg + \frac{1}{n}\sum_{j=1}^{N} w_{ij}\ll q_i a_j \gg, \tag{28}$$

$$-\frac{\partial}{\partial t}\ll a_i a_j \gg = 2\alpha \ll a_i a_j \gg - \overline{w_{ij}}(\ll r_i a_j \gg + \ll q_i a_j \gg) \\ -\overline{w_{ji}}(\ll a_i r_j \gg + \ll a_i q_j \gg), \tag{29}$$

$$-\frac{\partial}{\partial t}\ll r_i r_j \gg = -\alpha(\ll r_i a_j \gg + \ll a_i r_j \gg) + 2\beta \ll r_i r_j \gg + 2\overline{w_{ij}}\ll r_i r_j \gg, \tag{30}$$

$$-\frac{\partial}{\partial t}\ll a_i r_j \gg = -\alpha(\ll a_i a_j \gg - \ll a_i r_j \gg) + \beta \ll a_i r_j \gg + \overline{w_{ji}}\ll a_i r_j \gg \\ -\overline{w_{ij}}(\ll r_i r_j \gg + \ll q_i r_j \gg), \tag{31}$$

where

$$\overline{w_{lm}} = \frac{1}{n}[\sum_{k=1,(k\neq m)}^{N} w_{lk}\ll a_k \gg + w_{lm}]. \tag{32}$$

We note that the first moment dynamics obtained via the first approximation differs from that obtained from the second moment approximation. In the next section, we briefly examine this difference by comparing these approximations with Monte Carlo simulations.

## 4  COMPARISON WITH SIMULATIONS

In this section, we compare first and second moment approximations with Monte Carlo simulation of a one dimensional ring of three–state neurons. This was studied in a previous publication (Ohira and Cowan, 1993) for two–state neurons. As shown there, each three–state neuron in the ring interacts with its two neighbors.

More precisely, the Liouville operator is

$$
L = \alpha \sum_{i=1}^{N} (|a_i\rangle\langle a_i| - |r_i\rangle\langle a_i|) + \beta \sum_{i=1}^{N} (|r_i\rangle\langle r_i| - |q_i\rangle\langle r_i|) \tag{33}
$$

$$
+ \frac{1}{2} w_2 \sum_{i=1}^{N} (|r_i\rangle\langle r_i| - |a_i\rangle\langle r_i|)(|a_{i+1}\rangle\langle a_{i+1}| + |a_{i-1}\rangle\langle a_{i-1}|)
$$

$$
+ \frac{1}{2} w_1 \sum_{i=1}^{N} (|q_i\rangle\langle q_i| - |a_i\rangle\langle q_i|)(|a_{i+1}\rangle\langle a_{i+1}| + |a_{i-1}\rangle\langle a_{i-1}|)
$$

We now define the dynamical variables of interest as follows:

$$
\chi_a = \frac{1}{N} \sum_{i=1} \langle\!\langle a_i \rangle\!\rangle, \quad \chi_r = \frac{1}{N} \sum_{i=1} \langle\!\langle r_i \rangle\!\rangle, \quad \chi_q = \frac{1}{N} \sum_{i=1} \langle\!\langle q_i \rangle\!\rangle, \tag{34}
$$

$$
\eta_{aa} = \frac{1}{N} \sum_{i=1}^{N} \langle\!\langle a_i a_{i+1} \rangle\!\rangle, \quad \eta_{rr} = \frac{1}{N} \sum_{i=1}^{N} \langle\!\langle r_i r_{i+1} \rangle\!\rangle, \quad \eta_{ar} = \frac{1}{N} \sum_{i=1}^{N} \langle\!\langle a_i r_{i+1} \rangle\!\rangle. \tag{35}
$$

Then, for this network, the first moment approximation is given by

$$
\begin{aligned}
-\frac{\partial}{\partial t}\chi_a &= \alpha\chi - w_2\chi_a\chi_r - w_1\chi_q\chi_a, \\
-\frac{\partial}{\partial t}\chi_r &= -\alpha\chi - \beta\chi_r + w_2\chi_q\chi_a, \\
\chi_q &= 1 - \chi_a - \chi_r.
\end{aligned} \tag{36}
$$

The second moment approximation is given by

$$
\begin{aligned}
-\frac{\partial}{\partial t}\chi_a &= \alpha\chi - w_2\eta_{ar} - w_1(\chi_a - \eta_{ar} - \eta_{aa}), \\
-\frac{\partial}{\partial t}\chi_r &= -\alpha\chi - \beta\chi_r + w_2\eta_{ar}, \\
-\frac{\partial}{\partial t}\eta_{aa} &= 2\alpha\eta_{aa} - w_2\eta_{ar}(\chi_a + 1) - w_1(\chi_a + 1)(\chi_a - \eta_{ar} - \eta_{aa}), \\
-\frac{\partial}{\partial t}\eta_{ar} &= -\alpha(\eta_{aa} - \eta_{ar}) - \beta\eta_{ar} + \frac{1}{2}w_2\eta_{ar}(\chi_a + 1), \\
&\quad + \frac{1}{2}\eta_{rr}\chi_a + w_1\chi_a(\chi_r - \eta_{rr} - \eta_{ar}), \\
-\frac{\partial}{\partial t}\eta_{rr} &= -2\alpha\eta_{ar} - 2\beta\eta_{rr} + w_2\eta_{rr}\chi_a.
\end{aligned} \tag{37}
$$

Monte Carlo simulations of a ring of 10000 neurons were performed and compared with the first and second moment approximation predictions. We fixed the following parameters:

$$\alpha = 1.0, \quad \beta = 0.2, \quad w_1 = 0.01 \cdot w_0, \quad w_2 = 0.6 \cdot w_0 \qquad (38)$$

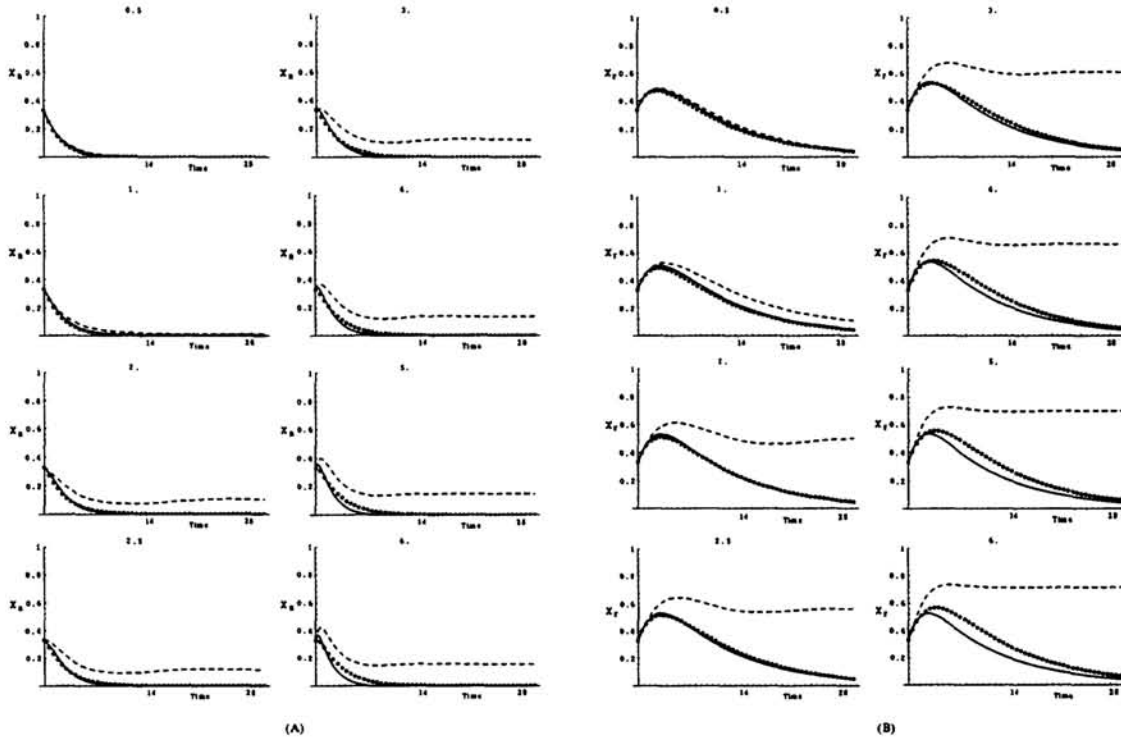

Figure 2: Comparison of Monte Carlo simulations (dots) with the first moment (dashed line) and the second moment (solid line) approximations for the three state case with the fraction of total active and refractory state variables $\chi_a$ (A) and $\chi_r$ (B). Each graph is labeled by the values of $w_0/\alpha$.

We varied $w_0$ and sampled the numerical dynamics of these parameters. Some comparisons are shown in Figure 2 for the time dependence of the total number of active and refractory state variables. We clearly see the improvement of the second over the first moment level approximation. More simulations with different parameter ranges remain to be explored.

## 5  CONCLUSION

We have introduced here a neural network master equation using the outer–product representation. In this representation, the extension from two to three–state neurons is transparent. We have taken advantage of this natural extension to analyse three–state networks. Even though the calculations involved are more intricate, we

have obtained results indicating that the second moment level approximation is significantly more accurate than the first moment level approximation. We also note that as in the two–state case, the first moment level approximation produces more activation than the simulation. Further analytical and theoretical investigations are needed to fully uncover the dynamics of three–state networks described by the master equation introduced above.

## Acknowledgements

This work was supported in part by the Robert R. McCormick fellowship at the University of Chicago, and in part by grant No. N0014-89-J-1099 from the US Department of the Navy, Office of Naval Research.

## References

Cowan JD (1991) Stochastic neurodynamics in Advances in Neural Information Processing Systems (D. S. Touretzky, R. P. Lippman, J. E. Moody, ed.), vol. 3, Morgan Kaufmann Publishers, San Mateo

Doi M (1976a) Second quantization representation for classical many-particle system. J. Phys. A: Math Gen. 9:1465–1477.

Doi M (1976b) Stochastic theory of diffusion-controlled reactions. J. Phys. A: Math. Gen. 9:1479.

Grassberger P, Scheunert M (1980) Fock-space methods for identical classical objects. Fortschritte der Physik 28:547

Meunier C, Hansel D, Verga A (1989) Information processing in three-state neural networks. J. Stat. Phys. 55:859

Ohira T, Cowan JD (1993) Master-equation approach to stochastic neurodynamics. Phys. Rev. E 48:2259

Ohira T, Cowan JD (1994) Feynman Diagrams for Stochastic Neurodynamics. In Proceedings of Fifth Australian Conference of Neural Networks, pp218-221

Sakurai JJ (1985) Modern Quantum Mechanics. Benjamin/Cummings, Menlo Park